# Product Analysis:
# Learning to model observations as products of hidden variables

Brendan J. Frey[1], Anitha Kannan[1], Nebojsa Jojic[2]

[1] Machine Learning Group, University of Toronto, www.psi.toronto.edu
[2] Vision Technology Group, Microsoft Research

## Abstract

Factor analysis and principal components analysis can be used to model linear relationships between observed variables and linearly map high-dimensional data to a lower-dimensional hidden space. In factor analysis, the observations are modeled as a linear combination of normally distributed hidden variables. We describe a nonlinear generalization of factor analysis, called "product analysis", that models the observed variables as a linear combination of *products* of normally distributed hidden variables. Just as factor analysis can be viewed as unsupervised linear regression on unobserved, normally distributed hidden variables, product analysis can be viewed as unsupervised linear regression on products of unobserved, normally distributed hidden variables. The mapping between the data and the hidden space is nonlinear, so we use an approximate variational technique for inference and learning. Since product analysis is a generalization of factor analysis, product analysis always finds a higher data likelihood than factor analysis. We give results on pattern recognition and illumination-invariant image clustering.

## 1 Introduction

Continuous-valued latent representations of observed feature vectors can be useful for pattern classification via Bayes rule, summarizing data sets, and producing low-dimensional representations of data for later processing.

Linear techniques, including principal components analysis (Jolliffe 1986), factor analysis (Rubin and Thayer 1982) and probabilistic principal components analysis (Tipping and Bishop 1999), model the input as a linear combination of hidden variables, plus sensor noise. The noise models are quite different in all 3 cases (see Tipping and Bishop (1999) for a discussion). For example, whereas factor analysis can account for different noise variances in the coordinates of the input, principal components analysis assumes that the noise variances are the same in different

input coordinates. Also, whereas factor analysis accounts for the sensor noise when estimating the combination weights, principal components analysis does not.

Often, the input coordinates are not linearly related, but instead the input vector is the result of a nonlinear generative process. In particular, data often can be accurately described as the *product* of unknown random variables. Examples include the combination of "style" and "content" (Tenenbaum and Freeman 1997), and the combination of a scalar light intensity and a reflectance image.

We introduce a generalization of factor analysis, called "product analysis", that performs maximum likelihood estimation to model the input as a linear combination of products of hidden variables. Although exact EM is not tractable because the hidden variables are nonlinearly related to the input, the form of the product analysis model makes it well-suited to a variational inference technique and a variational EM algorithm.

Other approaches to learning nonlinear representations include principal surface analysis (1984) and nonlinear autoencoders (Baldi and Hornik 1989; Diamantaras and Kung 1996), which minimize a reconstruction error when the data is mapped to the latent space and back; mixtures of linear models (Kambhatla and Leen 1994; Ghahramani and Hinton 1997; Tipping and Bishop 1999), which approximate nonlinear relationships using piece-wise linear patches; density networks (MacKay 1995), which use Markov chain Monte Carlo methods to learn potentially very complex density functions; generative topographic maps (Bishop, Svensén and Williams 1998), which use a finite set of fixed samples in the latent space for efficient inference and learning; and kernel principal components analysis (Schölkopf, Smola and Müller 1998), which finds principal directions in nonlinear functions of the input.

Our goals in developing product analysis is to introduce a technique that

- produces a density estimator of the data
- separates sensor noise from the latent structure
- learns a smooth, nonlinear map from the input to the latent space
- works for high-dimensional data and high-dimensional latent spaces
- is particularly well-suited to products of latent variables
- is computationally efficient

While none of the other approaches described above directly addresses all of these goals, product analysis does.

## 2   Factor analysis model

Of the three linear techniques described above, factor analysis has the simplest description as a generative model of the data. The input vector $\mathbf{x}$ is modeled using a vector of hidden variables $\mathbf{z}$. The hidden variables are independent and normally distributed with zero mean and unit variance:

$$p(\mathbf{z}) = \mathcal{N}(\mathbf{z}; \mathbf{0}, \mathbf{I}). \tag{1}$$

The input is modeled as a linear combination of the hidden variables, plus independent Gaussian noise:

$$p(\mathbf{x}|\mathbf{z}) = \mathcal{N}(\mathbf{x}; \mathbf{\Lambda}\mathbf{z}, \mathbf{\Psi}). \tag{2}$$

The model parameters are the factor loading matrix $\mathbf{\Lambda}$ and the diagonal matrix of sensor noise variances, $\mathbf{\Psi}$.

Factor analysis (c.f. (Rubin and Thayer 1982)) is the procedure for estimating $\mathbf{\Lambda}$ and $\mathbf{\Psi}$ using a training set. The marginal distribution over the input is $p(\mathbf{x}) = \mathcal{N}(\mathbf{x}; \mathbf{0}, \mathbf{\Lambda}\mathbf{\Lambda}^{\mathrm{T}} + \mathbf{\Psi})$, so factor analysis can be viewed as estimating a low-rank parameterization of the covariance matrix of the data.

## 3  Product analysis model

In the "product analyzer", the input vector $\mathbf{x}$ is modeled using a vector of hidden variables $\mathbf{z}$, which are independent and normally distributed with zero mean and unit variance:

$$p(\mathbf{z}) = \mathcal{N}(\mathbf{z}; \mathbf{0}, \mathbf{I}). \tag{3}$$

In factor analysis, the input is modeled as a linear combination of the hidden variables. In product analysis, the input is modeled as a linear combination of monomials in the hidden variables. The power of variable $z_k$ in monomial $i$ is $s_{ik}$. So, the $i$th monomial is

$$f_i(\mathbf{z}) = \prod_k z_k^{s_{ik}}. \tag{4}$$

Denoting the vector of $f_i(\mathbf{z})$'s by $\mathbf{f}(\mathbf{z})$, the density of the input given $\mathbf{z}$ is

$$p(\mathbf{x}|\mathbf{z}) = \mathcal{N}(\mathbf{x}; \mathbf{\Lambda}\mathbf{f}(\mathbf{z}), \mathbf{\Psi}). \tag{5}$$

The model parameters are $\mathbf{\Lambda}$ and the diagonal covariance matrix $\mathbf{\Psi}$. Here, we learn $\mathbf{\Lambda}$, maintaining the distribution over $\mathbf{z}$ constant. Alternatively, if $\mathbf{\Lambda}$ is known apriori, we can learn the distribution over $\mathbf{z}$, maintaining $\mathbf{\Lambda}$ to be fixed.

The matrix $\mathbf{S} = \{s_{ik}\}$ can be specified beforehand, estimated from the data using cross-validation, or averaged over in a Bayesian fashion. When $\mathbf{S} = \mathbf{I}$, $f(\mathbf{z}) = \mathbf{z}$ and the product analyzer simplifies to the factor analyzer. If, for some $i$, $s_{ik} = 0$, for all $k$, $f_i(\mathbf{z}) = 1$ and this monomial will account for a constant offset in the input.

## 4  Product analysis

Exact EM in the product analyzer is intractable, since the sufficient statistics require averaging over the posterior $p(\mathbf{z}|\mathbf{x})$, for which we do not have a tractable expression.

Instead, we use a variational approximation (Jordan et al. 1998), where for each training case, the posterior $p(\mathbf{z}|\mathbf{x})$ is approximated by a factorized Gaussian distribution $q(\mathbf{z})$ and the parameters of $q(\mathbf{z})$ are adjusted to make the approximation accurate. Then, the approximation $q(\mathbf{z})$ is used to compute the sufficient statistics for each training case in a generalized EM algorithm (Neal and Hinton 1993).

The $q$-distribution is specified by the variational parameters $\boldsymbol{\eta}$ and $\boldsymbol{\Phi}$:

$$q(\mathbf{z}) = \mathcal{N}(\mathbf{z}; \boldsymbol{\eta}, \boldsymbol{\Phi}), \tag{6}$$

where $\boldsymbol{\Phi}$ is a diagonal covariance matrix.

$q$ is optimized by minimizing the relative entropy (Kullback-Leibler divergence),

$$\mathcal{K} = \int_{\mathbf{z}} q(\mathbf{z}) \ln \frac{q(\mathbf{z})}{p(\mathbf{z}|\mathbf{x})}. \tag{7}$$

In fact, minimizing this entropy is equivalent to maximizing the following lower bound on the log-probability of the observation:

$$\mathcal{B} = \int_{\mathbf{z}} q(\mathbf{z}) \ln \frac{p(\mathbf{x}, \mathbf{z})}{q(\mathbf{z})} \leq \ln p(\mathbf{x}) \tag{8}$$

Pulling $\ln p(\mathbf{x})$ out of the integral, the bound can be expressed as

$$\mathcal{B} = \ln p(\mathbf{x}) - \int_{\mathbf{z}} q(\mathbf{z}) \ln \frac{q(\mathbf{z})}{p(\mathbf{z}|\mathbf{x})} = \ln p(\mathbf{x}) - \mathcal{K}. \tag{9}$$

Since $\ln p(\mathbf{x})$ does not directly depend on the variational parameters, maximizing $\mathcal{B}$ is equivalent to minimizing $\mathcal{K}$. Note that since $\mathcal{K} \geq 0$, $\mathcal{B} \leq \ln p(\mathbf{x})$. Using Lagrange multipliers, it is easy to show that the bound is maximized when $q(\mathbf{z}) = p(\mathbf{z}|\mathbf{x})$, in which case $\mathcal{K} = 0$ and $\mathcal{B} = \ln p(\mathbf{x})$.

Substituting the expressions for $p(\mathbf{z})$, $p(\mathbf{x}|\mathbf{z})$ and $q(\mathbf{z})$ into (8), and using the fact that $\mathbf{f}(\mathbf{z})^{\mathrm{T}} \boldsymbol{\Lambda}^{\mathrm{T}} \boldsymbol{\Psi}^{-1} \boldsymbol{\Lambda} \mathbf{f}(\mathbf{z}) = \mathrm{tr}\big(\mathbf{f}(\mathbf{z})^{\mathrm{T}} \boldsymbol{\Lambda}^{\mathrm{T}} \boldsymbol{\Psi}^{-1} \boldsymbol{\Lambda} \mathbf{f}(\mathbf{z})\big) = \mathrm{tr}\big(\boldsymbol{\Lambda}^{\mathrm{T}} \boldsymbol{\Psi}^{-1} \boldsymbol{\Lambda} \mathbf{f}(\mathbf{z})\mathbf{f}(\mathbf{z})^{\mathrm{T}}\big)$, we have

$$\begin{aligned}
\mathcal{B} = \frac{1}{2}\Big(& \ln|2\pi e \boldsymbol{\Phi}| - \ln|2\pi \boldsymbol{\Psi}| - \ln|2\pi \mathbf{I}| \\
& - \boldsymbol{\eta}^{\mathrm{T}}\boldsymbol{\eta} - \mathbf{x}^{\mathrm{T}}\boldsymbol{\Psi}^{-1}\mathbf{x} + 2\mathbf{x}^{\mathrm{T}}\boldsymbol{\Psi}^{-1}\boldsymbol{\Lambda}\mathrm{E}[\mathbf{f}(\mathbf{z})] + \mathrm{tr}\big(\boldsymbol{\Lambda}^{\mathrm{T}}\boldsymbol{\Psi}^{-1}\boldsymbol{\Lambda}\mathrm{E}[\mathbf{f}(\mathbf{z})\mathbf{f}(\mathbf{z})^{\mathrm{T}}]\big)\Big),
\end{aligned} \tag{10}$$

where $\mathrm{E}[]$ denotes an expectation with respect to $q(\mathbf{z})$.

The expectations are simplified as follows:

$$\mathrm{E}[f_i(\mathbf{z})] = \mathrm{E}[\prod_k z_k^{s_{ik}}] = \prod_k \mathrm{E}[z_k^{s_{ik}}] = \prod_k m_{s_{ik}}(\eta_k, \phi_k),$$

$$\mathrm{E}[f_i(\mathbf{z})f_j(\mathbf{z})] = \mathrm{E}[\prod_k z_k^{s_{ik}+s_{jk}}] = \prod_k \mathrm{E}[z_k^{s_{ik}+s_{jk}}] = \prod_k m_{s_{ik}+s_{jk}}(\eta_k, \phi_k), \tag{11}$$

where $m_n(\eta, \phi)$ is the $n$th moment under a Gaussian with mean $\eta$ and variance $\phi$. Closed forms for the $m_n(\eta, \phi)$ are found by setting derivatives of the Gaussian moment generating function to zero:

$$m_n(\eta, \phi) = \frac{\partial^n}{\partial t^n} \exp(\eta t + \phi^2 t^2/2)\Big|_{t=0}. \tag{12}$$

After substituting the closed forms for the moments, $\mathcal{B}$ is a polynomial in the $\eta_k$'s and the $\phi_k$'s. For each training case, $\mathcal{B}$ is maximized with respect to the $\eta_k$'s and the $\phi_k$'s using, $e.g.$, conjugate gradients. The model parameters $\boldsymbol{\Lambda}$ and $\boldsymbol{\Psi}$ that maximize the sum of the bounds for the training cases can be computed directly, since $\boldsymbol{\Psi}$ does not affect the solution for $\boldsymbol{\Lambda}$, $\mathcal{B}$ is quadratic in $\boldsymbol{\Lambda}$, and the optimal $\boldsymbol{\Psi}$ can be written in terms of $\boldsymbol{\Lambda}$ and the variational parameters.

If the power of each latent variable is restricted to be 0 or 1 in each monomial, $0 \leq s_{ik} \leq 1$, the above expressions simplify to

$$\mathrm{E}[f_i(\mathbf{z})] = \prod_k \eta_k^{s_{ik}}, \quad \mathrm{E}[f_i(\mathbf{z})f_j(\mathbf{z})] = \prod_k (\eta_k^{s_{ik}+s_{jk}} + \phi_k^{s_{ik}s_{jk}}). \tag{13}$$

In this case, we can directly maximize $\mathcal{B}$ with respect to each $\eta_k$ in turn, since $\mathcal{B}$ is quadratic in each $\eta_k$.

## 5   Experimental results:

**5.1   Classification results on the Wisconsin breast cancer database:**   We obtained results on using product analysis for classification of malignant and benign cancer using the breast cancer database provided by Dr. Wolberg from the Univ. of Wisconsin. Each observation in the database is characterized by nine cytological

a)                              b)          c)

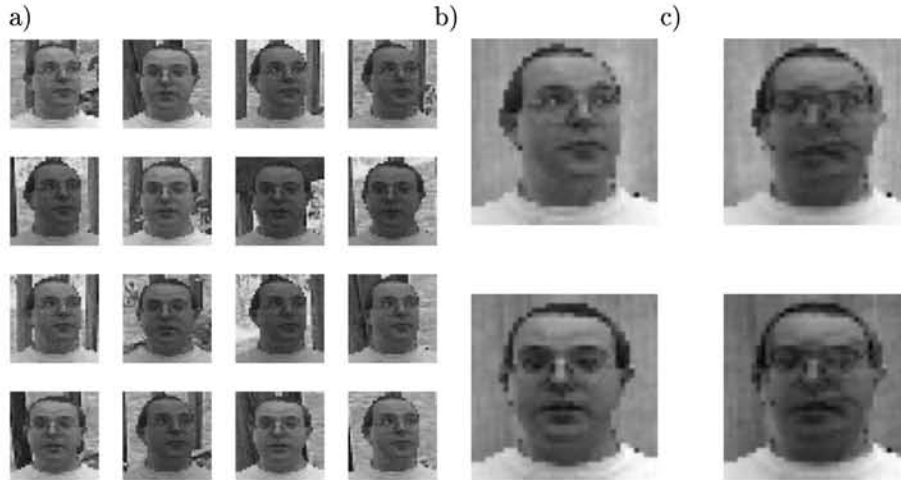

Figure 1: a) Data from training set. Mean images learned using b)product analysis c)mixture of gaussians

features, namely, lump thickness, uniformity of cell and shape, marginal adhesion, single epithelial cell size, bare nuclei, bland chromotin, normal nucleoli and mitoses. Each feature is assigned an integer between 1 and 10.

In their earlier work (Wolberg and Mangasarian 1990), the authors used linear programming for classification. The objective was to find a hyperplane that separates the classes of malignant and benign cancer. In the absence of a separating plane, average sum of misclassifications of each class is minimized.

Our approach is learn one density model for the benign feature vectors and a second density model for the malignant feature vectors and then use Bayes rule to classify an input vector. With separate models, classification involves assigning the observation to the model that provides the largest probability for occurrence of that observation as given by,

$$P(class|x) = \frac{P(x|class)P(class)}{P(x|benign)P(benign) + P(x|malignant)P(malignant)}$$

To compare with the result reported in (Wolberg and Mangasarian 1990), 4.1 % error rate on 369 instances, we used the same set for our learning scheme and found that the product analysis produced 4% misclassfication.

In addition, to compare the recognition rate of product analysis with the recognition rate of factor analysis, we divided the data set into 3 sets for training, validation and testing. The parameters of the model are learned using the training set, and tested on the validation set. This is repeated for 20 times, remembering the parameters that provided the best classification rate on the validation set. Finally, the parameters that provided the best performance on the validation set is used to classify the test set, only once. Since the data is limited, we perform this experimentation on 4 different random breakups of data into training, validation and test set. For product analysis model, we chose 3 hidden variables without optimization but for factor analysis, we chose the optimum number of factors. The average error rate on the 4 breakups was 5% using product analysis and 5.59% using factor analysis.

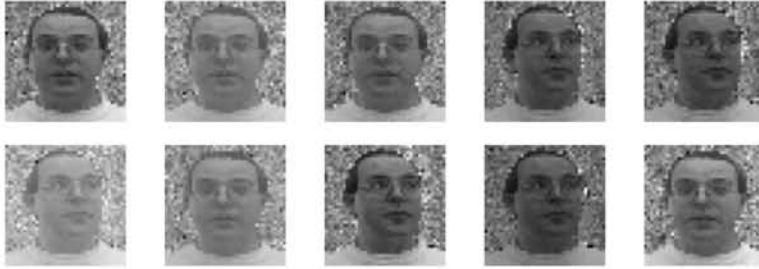

Figure 2: Images generated from the learned mixture of product analyzers

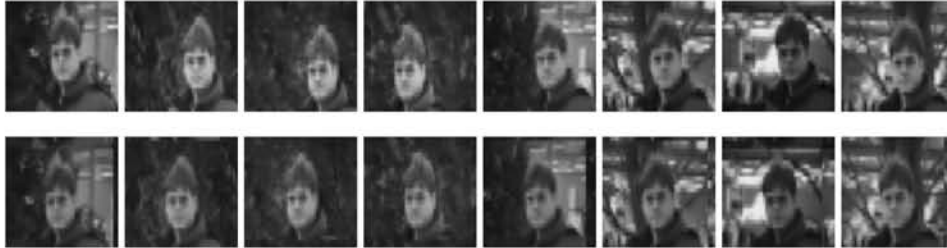

Figure 3: First row: Observation. Second row: corresponding image normalized for translation and lighting after lighting & transformation invariant model is learned

**5.2 Mixture of lighting invariant appearance models:** Often, objects are imaged under different illuminants. To learn an appearance model, we want to automatically remove the lighting effects and infer lighting-normalized images.

Since ambient light intensity and reflectances of patches on the object multiply to produce a lighting-affected image, we can model lighting-invariance using a product analyzer. $P(\mathbf{x}, \mathbf{z}) = P(\mathbf{x}|\mathbf{z})P(\mathbf{z})$, where $\mathbf{x}$ is the vector of pixel intensities of the observation, $z_1$ is the random variable describing the light intensity, and the remaining $z_i$ are the pixel intensities in the lighting normalized image. We learn the distribution over $\mathbf{z}$, where $\mathbf{f}(\mathbf{z}) = [z_1 z_2, z_1 z_3, ... z_1 z_{N+1}]^T$ and $\mathbf{\Lambda}$ is identity. By infering $z_1$, we can remove its effect on observation. The mixture model of product analyzer has joint distribution $\pi_c P(\mathbf{x}|\mathbf{z})P(\mathbf{z})$, where $\pi_c$ is the probability of each class. It can be used to infer various kinds of images (*e.g.* faces of different people) under different lighting conditions.

We trained this model on images with 2 different poses of the same person(Fig. 1a). The variation in the images is governed by change in pose, light, and background clutter. Fig. 1b and Fig. 1c compares the components learned using a mixture of product analyzers and a mixture of Gaussians. Due to limited variation in the pose and large variation in lighting, the mixture of gaussians is unable to extract the mean images. However, mixture of product analyzers is able to capture the distributions well. (Fig. 3).

**5.3 Transformation and lighting invariant appearance models:** Geometric transformations like shift and shearing can occur when scenes are imaged. Transformation invariant mixtures of Guassians and factor analyzers (Frey and Jojic 2002; Jojic et al. 2001) enable infering transformation-neutral image. Here, we add lighting-invariance to this framework enabling clustering based on interesting features such as pose, without concern for transformation and lighting effects.

# 6 Conclusions

We introduced a density model that explains observations as products of hidden variables and we presented a variational technique for inference and learning in this model. On the Wisonsin breast cancer data, we found that product analysis outperforms factor analysis, when used with Bayes rule for pattern classification. We also found that product analysis was able to separate the two hidden causes, lighting and image noise in noisy images with varying illumination and varying pose.

# References

Baldi, P. and Hornik, K. 1989. Neural networks and principal components analysis: Learning from examples without local minima. *Neural Networks*, 2:53–58.

Bishop, C. M., Svensén, M., and Williams, C. K. I. 1998. Gtm: the generative topographic mapping. *Neural Computation*, 10(1):215–235.

Diamantaras, K. I. and Kung, S. Y. 1996. *Principal Component Neural Networks*. Wiley, New York NY.

Frey, B. J. and Jojic, N. 2002. Transformation invariant clustering and linear component analysis. *IEEE Transactions on Pattern Analysis and Machine Intelligence*. To appear. Available at `http://www.cs.utoronto.ca/~frey`.

Ghahramani, Z. and Hinton, G. E. 1997. The EM algorithm for mixtures of factor analyzers. University of Toronto Technical Report CRG-TR-96-1.

Hastie, T. 1984. *Principal Curves and Surfaces*. Stanford University, Stanford CA. Doctoral dissertation.

Jojic, N., Simard, P., Frey, B. J., and Heckerman, D. 2001. Separating appearance from deformation. To appear in *Proceedings of the IEEE International Conference on Computer Vision*.

Jolliffe, I. T. 1986. *Principal Component Analysis*. Springer-Verlag, New York NY.

Jordan, M. I., Ghahramani, Z., Jaakkola, T. S., and Saul, L. K. 1998. An introduction to variational methods for graphical models. In Jordan, M. I., editor, *Learning in Graphical Models*. Kluwer Academic Publishers, Norwell MA.

Kambhatla, N. and Leen, T. K. 1994. Fast non-linear dimension reduction. In Cowan, J. D., Tesauro, G., and Alspector, J., editors, *Advances in Neural Information Processing Systems 6*, pages 152–159. Morgan Kaufmann, San Francisco CA.

MacKay, D. J. C. 1995. Bayesian neural networks and density networks. *Nuclear Instruments and Methods in Physics Research*, 354:73–80.

Neal, R. M. and Hinton, G. E. 1993. A new view of the EM algorithm that justifies incremental and other variants. Unpublished manuscript available over the internet by ftp at `ftp://ftp.cs.utoronto.ca/pub/radford/em.ps.Z`.

Rubin, D. and Thayer, D. 1982. EM algorithms for ML factor analysis. *Psychometrika*, 47(1):69–76.

Schölkopf, B., Smola, A., and Müller, K.-R. 1998. Nonlinear component analysis as a kernel eigenvalue problem. *Neural Computation*, 10:1299–1319.

Tenenbaum, J. B. and Freeman, W. T. 1997. Separating style and content. In Mozer, M. C., Jordan, M. I., and Petsche, T., editors, *Advances in Neural Information Processing Systems 9*. MIT Press, Cambridge MA.

Tipping, M. E. and Bishop, C. M. 1999. Mixtures of probabilistic principal component analyzers. *Neural Computation*, 11(2):443–482.

Wolberg, W. H. and Mangasarian, O. L. 1990. Multisurface method of pattern separation for medical diagnosis applied to breast cytology. In *Proceedings of the National Academy of Sciences*.
